# An Alternative Model for Mixtures of Experts

**Lei Xu**
Dept. of Computer Science, The Chinese University of Hong Kong
Shatin, Hong Kong, Email lxu@cs.cuhk.hk

**Michael I. Jordan**
Dept. of Brain and Cognitive Sciences
MIT
Cambridge, MA 02139

**Geoffrey E. Hinton**
Dept. of Computer Science
University of Toronto
Toronto, M5S 1A4, Canada

## Abstract

We propose an alternative model for mixtures of experts which uses a different parametric form for the gating network. The modified model is trained by the EM algorithm. In comparison with earlier models—trained by either EM or gradient ascent—there is no need to select a learning stepsize. We report simulation experiments which show that the new architecture yields faster convergence. We also apply the new model to two problem domains: piecewise nonlinear function approximation and the combination of multiple previously trained classifiers.

## 1 INTRODUCTION

For the *mixtures of experts* architecture (Jacobs, Jordan, Nowlan & Hinton, 1991), the EM algorithm decouples the learning process in a manner that fits well with the modular structure and yields a considerably improved rate of convergence (Jordan & Jacobs, 1994). The favorable properties of EM have also been shown by theoretical analyses (Jordan & Xu, in press; Xu & Jordan, 1994).

It is difficult to apply EM to some parts of the mixtures of experts architecture because of the nonlinearity of *softmax* gating network. This makes the maximiza-

tion with respect to the parameters in gating network nonlinear and analytically unsolvable even for the simplest generalized linear case. Jordan and Jacobs (1994) suggested a double-loop approach in which an inner loop of iteratively-reweighted least squares (IRLS) is used to perform the nonlinear optimization. However, this requires extra computation and the stepsize must be chosen carefully to guarantee the convergence of the inner loop.

We propose an alternative model for mixtures of experts which uses a different parametric form for the gating network. This form is chosen so that the maximization with respect to the parameters of the gating network can be handled analytically. Thus, a single-loop EM can be used, and no learning stepsize is required to guarantee convergence. We report simulation experiments which show that the new architecture yields faster convergence. We also apply the model to two problem domains. One is a piecewise nonlinear function approximation problem with smooth blending of pieces specified by polynomial, trigonometric, or other prespecified basis functions. The other is to combine classifiers developed previously—a general problem with a variety of applications (Xu, et al., 1991, 1992). Xu and Jordan (1993) proposed to solve the problem by using the mixtures of experts architecture and suggested an algorithm for bypassing the difficulty caused by the softmax gating networks. Here, we show that the algorithm of Xu and Jordan (1993) can be regarded as a special case of the single-loop EM given in this paper and that the single-loop EM also provides a further improvement.

## 2    MIXTURES OF EXPERTS AND EM LEARNING

The *mixtures of experts* model is based on the following conditional mixture:

$$P(y|x,\Theta) = \sum_{j=1}^{K} g_j(x,\nu)P(y|x,\theta_j),$$

$$P(y|x,\theta_j) = \frac{1}{(2\pi)^{n/2}|\Gamma_j|^{1/2}} \exp\{-\frac{1}{2}[y-f_j(x,w_j)]^T\Gamma_j^{-1}[y-f_j(x,w_j)]\} \quad (1)$$

where $x \in R^n$, and $\Theta$ consists of $\nu, \{\theta_j\}_1^K$, and $\theta_j$ consists of $\{w_j\}_1^K, \{\Gamma_j\}_1^K$. The vector $f_j(x,w_j)$ is the output of the $j$-th expert net. The scalar $g_j(x,\nu), j = 1, \cdots, K$ is given by the *softmax* function:

$$g_j(x,\nu) = e^{\beta_j(x,\nu)}/\sum_i e^{\beta_i(x,\nu)}. \quad (2)$$

In this equation, $\beta_j(x,\nu), j = 1, \cdots, K$ are the outputs of the *gating network*.

The parameter $\Theta$ is estimated by Maximum Likelihood (ML), where the log likelihood is given by $L = \sum_t \ln P(y^{(t)}|x^{(t)},\Theta)$. The ML estimate can be found iteratively using the EM algorithm as follows. Given the current estimate $\Theta^{(k)}$, each iteration consists of two steps.

(1) E-step. For each pair $\{x^{(t)}, y^{(t)}\}$, compute $h_j^{(k)}(y^{(t)}|x^{(t)}) = P(j|x^{(t)}, y^{(t)})$, and then form a set of objective functions:

$$Q_j^e(\theta_j) = \sum_t h_j^{(k)}(y^{(t)}|x^{(t)}) \ln P(y^{(t)}|x^{(t)},\theta_j), \quad j = 1, \cdots, K;$$

$$Q^g(\nu) = \sum_t \sum_j h_j^{(k)}(y^{(t)}|x^{(t)}) \ln g_j^{(k)}(x^{(t)}, \nu^{(k)}). \tag{3}$$

(2). M-step. Find a new estimate $\Theta^{(k+1)} = \{\{\theta_j^{(k+1)}\}_{j=1}^K, \nu^{(k+1)}\}$ with:

$$\theta_j^{(k+1)} = \arg\max_{\theta_j} Q_j^e(\theta_j), j = 1, \cdots, K; \quad \nu^{(k+1)} = \arg\max_{\nu} Q^g(\nu). \tag{4}$$

In certain cases, for example when $f_j(x, w_j)$ is linear in the parameters $w_j$, $\max_{\theta_j} Q_j^e(\theta_j)$ can be solved by solving $\partial Q_j^e/\partial \theta_j = 0$. When $f_j(x, w_j)$ is nonlinear with respect to $w_j$, however, the maximization cannot be performed analytically.

Moreover, due to the nonlinearity of *softmax*, $\max_\nu Q^g(\nu)$ cannot be solved analytically in any case. There are two possibilities for attacking these nonlinear optimization problems. One is to use a conventional iterative optimization technique (e.g., gradient ascent) to perform one or more inner-loop iterations. The other is to simply find a new estimate such that $Q_j^e(\theta_j^{(k+1)}) \geq Q_j^e(\theta_j^{(k)})$, $Q^g(\nu^{(k+1)}) \geq Q^g(\nu^{(k)})$. Usually, the algorithms that perform a full maximization during the M step are referred as "EM" algorithms, and algorithms that simply increase the $Q$ function during the M step as "GEM" algorithms. In this paper we will further distinguish between EM algorithms requiring and not requiring an iterative inner loop by designating them as *double-loop EM* and *single-loop EM* respectively.

Jordan and Jacobs (1994) considered the case of linear $\beta_j(x, \nu) = \nu_j^T[x, 1]$ with $\nu = [\nu_1, \cdots, \nu_K]$ and semi-linear $f_j(w_j^T[x, 1])$ with nonlinear $f_j(.)$. They proposed a double-loop EM algorithm by using the IRLS method to implement the inner-loop iteration. For more general nonlinear $\beta_j(x, \nu)$ and $f_j(x, \theta_j)$, Jordan and Xu (in press) showed that an extended IRLS can be used for this inner loop. It can be shown that IRLS and the extension are equivalent to solving eq. (3) by the so-called *Fisher Scoring* method.

## 3   A NEW GATING NET AND A SINGLE-LOOP EM

To sidestep the need for a nonlinear optimization routine in the inner loop of the EM algorithm, we propose the following modified gating network:

$$g_j(x, \nu) = \alpha_j P(x|\nu_j)/\sum_i \alpha_i P(x|\nu_i), \quad \sum_j \alpha_j = 1, \alpha_j \geq 0,$$
$$P(x|\nu_j) = a_j(\nu_j)^{-1} b_j(x) \exp\{c_j(\nu_j)^T t_j(x)\} \tag{5}$$

where $\nu = \{\alpha_j, \nu_j, j = 1, \cdots, K\}$, $t_j(x)$ is a vector of sufficient statistics, and the $P(x|\nu_j)$'s are density functions from the exponential family. The most common example is the Gaussian:

$$P(x|\nu_j) = \frac{1}{(2\pi)^{n/2}|\Sigma_j|^{1/2}} \exp\{-\frac{1}{2}(x - m_j)^T \Sigma_j^{-1}(x - m_j)\}, \tag{6}$$

In eq. (5), $g_j(x, \nu)$ is actually the posterior probability $P(j|x)$ that $x$ is assigned to the partition corresponding to the $j$-th expert net, obtained from Bayes' rule:

$$g_j(x, \nu) = P(j|x) = \alpha_j P(x|\nu_j)/P(x, \nu), \quad P(x, \nu) = \sum_i \alpha_i P(x|\nu_i). \tag{7}$$

Inserting this $g_j(x, \nu)$ into the model eq. (1), we get

$$P(y|x, \Theta) = \sum_j \frac{\alpha_j P(x|\nu_j)}{P(x, \nu)} P(y|x, \theta_j). \tag{8}$$

If we do ML estimation directly on this $P(y|x, \Theta)$ and derive an EM algorithm, we again find that the maximization $\max_\nu Q^g(\nu)$ cannot be solved analytically. To avoid this difficulty, we rewrite eq. (8) as:

$$P(y, x) = P(y|x, \Theta) P(x, \nu) = \sum_j \alpha_j P(x|\nu_j) P(y|x, \theta_j). \tag{9}$$

This suggests an asymmetrical representation for the joint density. We accordingly perform ML estimation based on $L' = \sum_t \ln P(y^{(t)}, x^{(t)})$ to determine the parameters $\alpha_j, \nu_j, \theta_j$ of the gating net and the expert nets. This can be done by the following EM algorithm:

(1) E-step. Compute

$$h_j^{(k)}(y^{(t)}|x^{(t)}) = \frac{\alpha_j^{(k)} P(x^{(t)}|\nu_j^{(k)}) P(y^{(t)}|x^{(t)}, \theta_j^{(k)})}{\sum_i \alpha_i^{(k)} P(x^{(t)}|\nu_i^{(k)}) P(y^{(t)}|x^{(t)}, \theta_j^{(k)})}; \tag{10}$$

Then let $Q_j^e(\theta_j), j = 1, \cdots, K$ be the same as given in eq. (3), and decompose $Q^g(\nu)$ further into

$$Q_j^g(\nu_j) = \sum_t h_j^{(k)}(y^{(t)}|x^{(t)}) \ln P(x^{(t)}|\nu_j), \quad j = 1, \cdots, K;$$

$$Q^\alpha = \sum_t \sum_j h_j^{(k)}(y^{(t)}|x^{(t)}) \ln \alpha_j, \quad \text{with } \alpha = \{\alpha_1, \cdots, \alpha_K\}. \tag{11}$$

(2). M-step. Find a new estimate for $j = 1, \cdots, K$

$$\theta_j^{(k+1)} = \arg\max_{\theta_j} Q_j^e(\theta_j), \quad \nu_j^{(k+1)} = \arg\max_{\nu_j} Q_j^g(\nu_j),$$
$$\alpha^{(k+1)} = \arg\max_\alpha Q^\alpha, \quad s.t. \sum_j \alpha_j = 1. \tag{12}$$

The maximization for the expert nets is the same as in eq. (4). However, for the gating net the maximization now becomes analytically solvable as long as $P(x|\nu_j)$ is from the exponential family. That is, we have:

$$\nu_j^{(k+1)} = \frac{\sum_t h_j^{(k)}(y^{(t)}|x^{(t)}) t_j(x^{(t)})}{\sum_t h_j^{(k)}(y^{(t)}|x^{(t)})}, \quad \alpha_j^{(k+1)} = \frac{1}{N} \sum_t h_j^{(k)}(y^{(t)}|x^{(t)}). \tag{13}$$

In particular, when $P(x|\nu_j)$ is a Gaussian density, the update becomes:

$$m_j^{(k+1)} = \frac{1}{\sum_t h_j^{(k)}(y^{(t)}|x^{(t)})} \sum_t h_j^{(k)}(y^{(t)}|x^{(t)}) x^{(t)},$$

$$\Sigma_j^{(k+1)} = \frac{1}{\sum_t h_j^{(k)}(y^{(t)}|x^{(t)})} \sum_t h_j^{(k)}(y^{(t)}|x^{(t)})[x^{(t)} - m_j^{(k+1)}][x^{(t)} - m_j^{(k+1)}]^T. \tag{14}$$

Two issues deserve to be emphasized further:

(1) The gating nets eq. (2) and eq. (5) become identical when $\beta_j(x, \nu) = \ln \alpha_j + \ln b_j(x) + c_j(\nu_j)^T t_j(x) - \ln a_j(\nu_j)$. In other words, the gating net in eq. (5) explicitly uses this function family instead of the function family defined by a multilayer feedforward network.

(2) It follows from eq. (9) that $\max \ln P(y, x|\Theta) = \max [\ln P(y|x, \Theta) + \ln P(x|\nu)]$. So, the solution given by eqs. (10) through (14) is actually different from the one given by the original eqs. (3) and (4). The former tries to model both the mapping from $x$ to $y$ and the input $x$, while the latter only models the mapping from $x$ and $y$. In fact, here we learn the parameters of the gating net and the expert nets via an asymmetrical representation eq. (9) of the joint density $P(y, x)$ which includes $P(y|x)$ implicitly. However, in the testing phase, the total output still follows eq. (8).

## 4    PIECEWISE NONLINEAR APPROXIMATION

The simple form $f_j(x, w_j) = w_j^T[x, 1]$ is not the only case to which single-loop EM applies. Whenever $f_j(x, w_j)$ can be written in a form linear in the parameters:

$$f_j(x, w_j) = \sum_i w_{i,j} \phi_{i,j}(x) + w_{0,j} = w_j^T[\phi_j(x), 1], \qquad (15)$$

where $\phi_{i,j}(x)$ are prespecified basis functions, $\max_{\theta_j} Q_j^e(\theta_j), j = 1, \cdots, K$ in eq. (3) is still a weighted least squares problem that can be solved analytically. One useful special case is when $\phi_{i,j}(x)$ are canonical polynomial terms $x_1^{r_1} \cdots x_d^{r_d}$, $r_i \geq 0$. In this case, the mixture of experts model implements piecewise polynomial approximations. Another case is that $\phi_{i,j}(x)$ is $\prod_i \sin_i^r(j\pi x_1) \cos_i^r(j\pi x_1), r_i \geq 0$, in which case the mixture of experts implements piecewise trigonometric approximations.

## 5    COMBINING MULTIPLE CLASSIFIERS

Given pattern classes $C_i, i = 1, \cdots, M$, we consider classifiers $e_j$ that for each input $x$ produce an output $P_j(y|x)$:

$$P_j(y|x) = [p_j(1|x), \cdots, p_j(M|x)], \quad p_j(i|x) \geq 0, \sum_i p_j(i|x) = 1. \qquad (16)$$

The problem of *Combining Multiple Classifiers (CMC)* is to combine these $P_j(y|x)$'s to give a combined estimate of $P(y|x)$. Xu and Jordan (1993) proposed to solve CMC problems by regarding the problem as a special example of the mixture density problem eq. (1) with the $P_j(y|x)$'s known and only the gating net $g_j(x, \nu)$ to be learned. In Xu and Jordan (1993), one problem encountered was also the nonlinearity of *softmax* gating networks, and an algorithm was proposed to avoid the difficulty.

Actually, the single-loop EM given by eq. (10) and eq. (13) can be directly used to solve the CMC problem. In particular, when $P(x|\nu_j)$ is Gaussian, eq. (13) becomes eq. (14). Assuming that $\alpha_1 = \cdots = \alpha_K$ in eq. (7), eq. (10) becomes

$h_j^{(k)}(y^{(t)}|x^{(t)}) = P(x^{(t)}|\nu_j^{(k)})P(y^{(t)}|x^{(t)})/\sum_i P(x^{(t)}|\nu_i^{(k)})P(y^{(t)}|x^{(t)})$. If we divide both the numerator and denominator by $\sum_i P(x^{(t)}|\nu_i^{(k)})$, we get $h_j^{(k)}(y^{(t)}|x^{(t)}) = g_j(x,\nu)P(y^{(t)}|x^{(t)})/\sum_i g_j(x,\nu)P(y^{(t)}|x^{(t)})$. Comparing this equation with eq. (7a) in Xu and Jordan (1993), we can see that the two equations are actually the same. Despite the different notation, $\alpha_j(\mathbf{x})$ and $P_j(\vec{y}^{(t)}|\mathbf{x}^{(t)})$ in Xu and Jordan (1993) are the same as $g_j(x,\nu)$ and $P(y^{(t)}|x^{(t)})$ in Section 3. So the algorithm of Xu and Jordan (1993) is a special case of the single-loop EM given in Section 3.

## 6   SIMULATION RESULTS

We compare the performance of the EM algorithm presented earlier with the model of mixtures of experts presented by Jordan and Jacobs (1994). As shown in Fig. 1(a), we consider a mixture of experts model with $K = 2$. For the expert nets, each $P(y|x,\theta_j)$ is Gaussian given by eq. (1) with linear $f_j(x,w_j) = w_j^T[x,1]$. For the new gating net, each $P(x,\nu_j)$ in eq. (5) is Gaussian given by eq. (6). For the old gating net eq. (2), $\beta_1(x,\nu) = 0$ and $\beta_2(x,\nu) = \nu^T[x,1]$. The learning speeds of the two are significantly different. The new algorithm takes k=15 iterations for the log-likelihood to converge to the value of $-1271.8$. These iterations require about $1,351,383$ MATLAB *flops*. For the old algorithm, we use the IRLS algorithm given in Jordan and Jacobs (1994) for the inner loop iteration. In experiments, we found that it usually took a large number of iterations for the inner loop to converge. To save computations, we limit the maximum number of iterations by $\tau_{max} = 10$. We found that this saved computation without obviously influencing the overall performance. From Fig. 1(b), we see that the outer loop converges in about 16 iterations. Each inner loop takes 290498 *flops* and the entire process requires $5,312,695$ *flops*. So, we see that the new algorithm yields a speedup of about $4,648,608/1,441,475 = 3.9$. Moreover, no external adjustment is needed to ensure the convergence of the new algorithm. But for the old one the direct use of IRLS can make the inner loop diverge and we need to appropriately rescale the updating stepsize of IRLS.

Figs. 2(a) and (b) show the results of a simulation of a piecewise polynomial approximation problem utilizing the approach described in Section 4. We consider a mixture of experts model with $K = 2$. For expert nets, each $P(y|x,\theta_j)$ is Gaussian given by eq. (1) with $f_j(x,w_j) = w_{3,j}x^3 + w_{2,j}x^2 + w_{1,j}x + w_{0,j}$. In the new gating net eq. (5), each $P(x,\nu_j)$ is again Gaussian given by eq. (6). We see that the higher order nonlinear regression has been fit quite well.

For multiple classifier combination, the problem and data are the same as in Xu and Jordan (1993). Table 1 shows the classification results. *Com-old* and *Com-new* denote the method given in in Xu and Jordan (1993) and in Section 5 respectively. We see that both improve the classification rate of each individual network considerably and that $Com - new$ improves on $Com - old$.

|              | Classifer $e_1$ | Classifer $e_1$ | Com − old | Com − new |
|--------------|-----------------|-----------------|-----------|-----------|
| *Training set* | 89.9%           | 93.3%           | 98.6%     | 99.4%     |
| *Testing set*  | 89.2%           | 92.7%           | 98.0%     | 99.0%     |

Table 1 A comparison of the correct classification rates

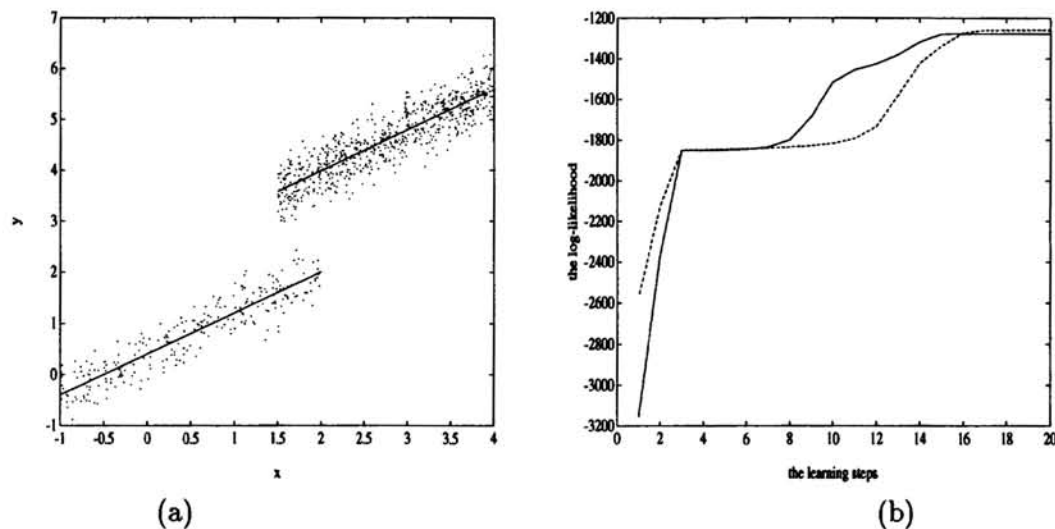

|     |     |
| :-: | :-: |
| (a) | (b) |

Figure 1: (a) 1000 samples from $y = a_1 x + a_2 + \varepsilon, a_1 = 0.8, a_2 = 0.4, x \in [-1, 1.5]$ with prior $\alpha_1 = 0.25$ and $y = a_1' x + a_2' + \varepsilon, a_1' = 0.8, a_2 =' 2.4, x \in [1, 4]$ with prior $\alpha_2 = 0.75$, where $x$ is uniform random variable and $z$ is from Gaussian $N(0, 0.3)$. The two lines through the clouds are the estimated models of two expert nets. The fits obtained by the two learning algorithms are almost the same. (b) The evolution of the log-likelihood. The solid line is for the modified learning algorithm. The dotted line is for the original learning algorithm (the outer loop iteration).

## 7 REMARKS

Recently, Ghahramani and Jordan (1994) proposed solving function approximation problems by using a mixture of Gaussians to estimate the joint density of the input and output (see also Specht, 1991; Tresp, et al., 1994). In the special case of linear $f_j(x, w_j) = w_j^T[x, 1]$ and Gaussian $P(x|\nu_j)$ with equal priors, the method given in Section 3 provides the same result as Ghahramani and Jordan (1994) although the parameterizations of the two methods are different. However, the method of this paper also applies to nonlinear $f_j(x, w_j) = w_j^T[\phi_j(x), 1]$ for piecewise nonlinear approximation or more generally $f_j(x, w_j)$ that is nonlinear with respect to $w_j$, and applies to cases in which $P(y, x|\nu_j, \theta_j)$ is not Gaussian, as well as the case of combining multiple classifiers. Furthermore, the methods proposed in Sections 3 and 4 can also be extended to the hierarchical mixture of experts architecture (Jacobs & Jordan, 1994) so that single-loop EM can be used to facilitate its training.

### References

Ghahramani, Z., & Jordan, M.I. (1994). Function approximation via density estimation using the EM approach. In Cowan, J.D., Tesauro, G., and Alspector, J., (Eds.), *Advances in NIPS 6*. San Mateo, CA: Morgan Kaufmann.

Jacobs, R.A., Jordan, M.I., Nowlan, S.J., & Hinton, G.E. (1991). Adaptive mixtures of local experts. *Neural Computation, 3*, 79-87.

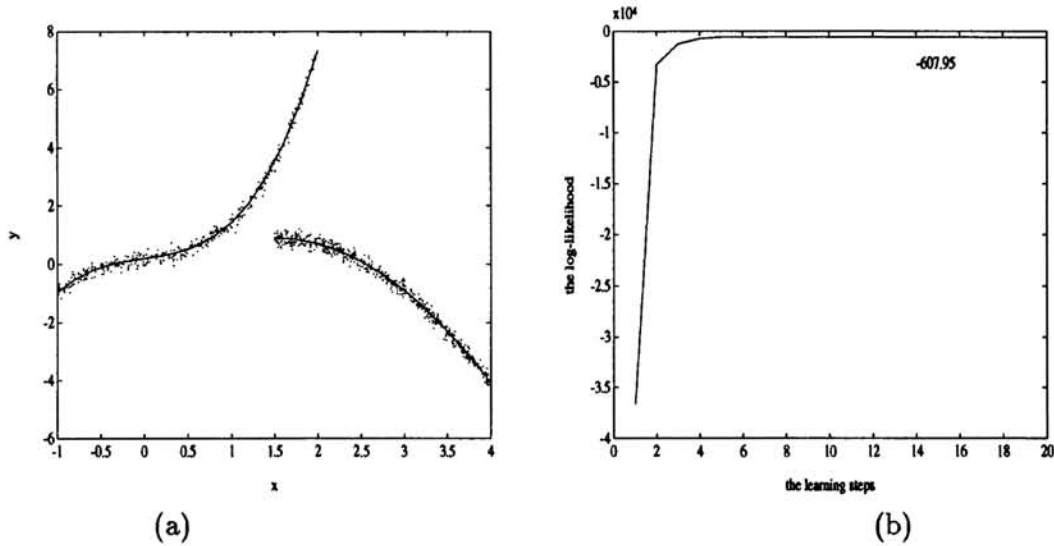

Figure 2: Piecewise 3rd polynomial approximation. (a) 1000 samples from $y = a_1x^3 + a_3x + a_4 + \varepsilon$, $x \in [-1, 1.5]$ with prior $\alpha_1 = 0.4$ and $y = a'_2x^2 + a'_3x^2 + a'_4 + \varepsilon$, $x \in [1, 4]$ with prior $\alpha_2 = 0.6$, where $x$ is uniform random variable and $z$ is from Gaussian $N(0, 0.15)$. The two curves through the clouds are the estimated models of two expert nets. (b) The evolution of the log-likelihood.

Jordan, M.I., & Jacobs, R.A. (1994). Hierarchical mixtures of experts and the EM algorithm. *Neural Computation, 6*, 181-214.

Jordan, M.I., & Xu, L. (in press). Convergence results for the EM approach to mixtures-of-experts architectures. *Neural Networks*.

Specht, D. (1991). A general regression neural network. *IEEE Trans. Neural Networks, 2*, 568-576.

Tresp, V., Ahmad, S., and Neuneier, R. (1994). Training neural networks with deficient data. In Cowan, J.D., Tesauro, G., & Alspector, J., (Eds.), *Advances in NIPS 6*, San Mateo, CA: Morgan Kaufmann.

Xu, L., Krzyzak A., & Suen, C.Y. (1991). Associative switch for combining multiple classifiers. *Proc. of 1991 IJCNN*, Vol. I. Seattle, 43-48.

Xu, L., Krzyzak A., & Suen, C.Y. (1992). Several methods for combining multiple classifiers and their applications in handwritten character recognition. *IEEE Trans. on SMC*, Vol. SMC-22, 418-435.

Xu, L., & Jordan, M.I. (1993). EM Learning on a generalized finite mixture model for combining multiple classifiers. *Proceedings of World Congress on Neural Networks*, Vol. IV. Portland, OR, 227-230.

Xu, L., & Jordan, M.I. (1994). On convergence properties of the EM algorithm for Gaussian mixtures. Submitted to *Neural Computation*.
